# A Constructive Learning Algorithm for Discriminant Tangent Models

**Diego Sona**   **Alessandro Sperduti**   **Antonina Starita**
Dipartimento di Informatica, Università di Pisa
Corso Italia, 40, 56125 Pisa, Italy
email: {sona,perso,starita}di.unipi.it

## Abstract

To reduce the computational complexity of classification systems using tangent distance, Hastie et al. (HSS) developed an algorithm to devise rich models for representing large subsets of the data which computes automatically the "best" associated tangent subspace. Schwenk & Milgram proposed a discriminant modular classification system (*Diabolo*) based on several autoassociative multilayer perceptrons which use tangent distance as error reconstruction measure.

We propose a gradient based constructive learning algorithm for building a tangent subspace model with discriminant capabilities which combines several of the the advantages of both HSS and Diabolo: devised tangent models hold discriminant capabilities, space requirements are improved with respect to HSS since our algorithm is discriminant and thus it needs fewer prototype models, dimension of the tangent subspace is determined automatically by the constructive algorithm, and our algorithm is able to learn new transformations.

## 1  Introduction

Tangent distance is a well known technique used for transformation invariant pattern recognition. State-of-the-art accuracy can be achieved on an isolated handwritten character task using tangent distance as the classification metric within a nearest neighbor algorithm [SCD93]. However, this approach has a quite high computational complexity, owing to the inefficient search and large number of Euclidean and tangent distances that need to be calculated. Different researchers have shown how such time complexity can be reduced [Sim94, SS95] at the cost of increased space complexity.

A different approach to the problem was used by Hastie et al. [HSS95] and Schwenk & Milgram [SM95b, SM95a]. Both of them used learning algorithms for reducing the classification time and space requirements, while trying to preserve the same accuracy. Hastie et al. [HSS95] developed rich models for representing large subsets of the prototypes. These models are learned from a training set through a Singular Value Decomposition based algorithm which minimizes the average 2-sided tangent distance from a subset of the training images. A nice feature of this algorithm is that it computes automatically the "best" tangent subspace associated with the prototypes. Schwenk & Milgram [SM95b] proposed a modular classification system (*Diabolo*) based on several autoassociative multilayer perceptrons which use tangent distance as the error reconstruction measure. This original model was then improved by adding discriminant capabilities to the system [SM95a].

Comparing Hastie et al. algorithm (HSS) versus the discriminant version of Diabolo, we observe that: Diabolo seems to require less memory than HSS, however, learning is faster in HSS; Diabolo is discriminant while HSS is not; the number of hidden units to be used in Diabolo's autoassociators must be decided heuristically through a trial and error procedure, while the dimension of the tangent subspaces in HSS can be controlled more easily; Diabolo uses predefined transformations, while HSS is able to learn new transformations (like style transformations).

In this paper, we introduce the *tangent distance neuron* (TD-neuron), which implements the 1-sided version of the tangent distance, and we devise a gradient based constructive learning algorithm for building a tangent subspace model with discriminant capabilities. In this way, we are able to combine the advantages of both HSS and Diabolo: the model holds discriminant capabilities, learning is just a bit slower than HSS, space requirements are improved with respect to HSS since the TD-neuron is discriminant and thus it needs fewer prototype models, the dimension of the tangent subspace is determined automatically by the constructive algorithm, and TD-neuron is able to learn new transformations.

## 2  Tangent Distance

In several pattern recognition problems Euclidean distance fails to give a satisfactory solution since it is unable to account for invariant transformations of the patterns. Simard et al. [SCD93] suggested dealing with this problem by generating a parameterized 7-dimensional manifold for each image, where each parameter accounts for one such invariance. The underlying idea consists in approximating the considered transformations locally through a linear model.

For the sake of exposition, consider rotation. Given a digitalized image $X_i$ of a pattern $i$, the rotation operation can be approximated by $\tilde{X}_i(\theta) = X_i + T_{X_i}\theta$, where $\theta$ is the rotation angle, and $T_{X_i}$ is the tangent vector to the rotation curve generated by the rotation operator for $X_i$. The tangent vector $T_{X_i}$ can easily be computed by finite difference. Now, instead of measuring the distance between two images as $D(X_i, X_j) = \|X_i - X_j\|$ for any norm $\|\cdot\|$, Simard et al. proposed using the *tangent distance* $D_T(X_i, X_j) = \min_{\theta_i, \theta_j} \|\tilde{X}_i(\theta_i) - \tilde{X}_j(\theta_j)\|$.

If $k$ types of transformations are considered, there will be $k$ different tangent vectors per pattern. If $\|\cdot\|$ is the Euclidean norm, computing the tangent distance is a simple least-squares problem. A solution for this problem[1] can be found in Simard et al. [SCD93], where the authors used $D_T$ to drive a 1-NN classification rule.

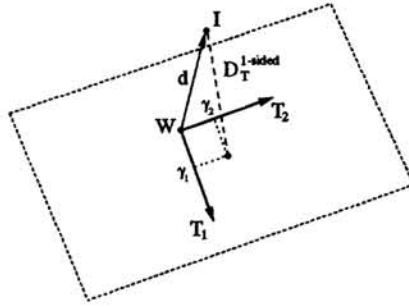

Figure 1: Geometric interpretation of equation 1. Note that $net = (D_T^{1-sided})^2$.

Unfortunately, 1-NN is expensive. To reduce the complexity of the above approach, Hastie et al. [HSS95] proposed an algorithm for the generation of rich models representing large subsets of patterns. This algorithm computes for each class a prototype (the *centroid*), and an associated subspace (described by the tangent vectors), such that the total tangent distance of the centroid with respect to the prototypes in the training set is minimised. Note that the associated subspace is not predefined as in the case of standard tangent distance, but is computed on the basis of the training set.

## 3    Tangent Distance Neuron

In this section we define the Tangent Distance neuron (TD-neuron), which is the computational model studied in this paper. A TD-neuron is characterized by a set of $n + 1$ vectors, of the same dimension as the input vectors (in our case, images). One of these vectors, $W$ is used as reference vector (*centroid*), while the remaining vectors, $T_i$ $(i = 1, \ldots, n)$, are used as *tangent* vectors. Moreover, the set of *tangent* vectors constitutes an *ortho-normal basis*.

Given an input vector $I$ the input net of the TD-neuron is computed as the square of the 1-sided tangent distance between $I$ and the tangent model $\{W, T_1, \ldots, T_n\}$ (see Figure 1)

$$net = \| \underbrace{I - W}_{d} \|^2 - \sum_{i=1}^{n} [(I - W)^t T_i]^2 = d^t d - \sum_{i=1}^{n} [\underbrace{d^t T_i}_{\gamma_i}]^2 \qquad (1)$$

where we have used the fact that the *tangent* vectors constitute an ortho-normal basis. For the sake of notation, $d$ denotes the difference between the input pattern and the centroid, and the projection of $d$ over the i-th tangent vector is denoted by $\gamma_i$. Note that, by definition, $net$ is non-negative.

The output $o$ of the TD-neuron is then computed by transforming the $net$ through a nonlinear monotone function $f$. In our experiments, we have used the following function

$$o = f(\alpha, net) = \frac{1}{1 + \alpha \, net} \qquad (2)$$

where $\alpha$ controls the steepness of the function. Note that $o$ is positive since $net$ is always positive and within the range $(0, 1]$.

## 4 Learning

The TD-neuron can be trained to discriminate between patterns belonging to two different classes through a gradient descent technique. Thus, given a training set $\{(I_1, t_1), \ldots, (I_N, t_N)\}$, where $t_i \in \{0, 1\}$ is the i-th desired output, and $N$ is the total number of patterns in the training set, we can define the error function as

$$E = \frac{1}{2} \sum_{k=1}^{N} (t_k - o_k)^2 \tag{3}$$

where $o_k$ is the output of the TD-neuron for the k-th input pattern.

Using equations (1-2), it is trivial to compute the changes for the tangent vectors, the centroid and $\alpha$:

$$\Delta T_i = -\eta \left( \frac{\delta E}{\delta T_i} \right) = 2\alpha\eta \sum_{k=1}^{N} (t_k - o_k) \, o_k^2 \gamma_{ik} d_k \tag{4}$$

$$\Delta W = -\eta \left( \frac{\delta E}{\delta W} \right) = 2\alpha\eta \sum_{k=1}^{N} (t_k - o_k) \, o_k^2 (d_k - \sum_{i=1}^{n} \gamma_{ik} T_i) \tag{5}$$

$$\Delta \alpha = -\eta_\alpha \left( \frac{\delta E}{\delta \alpha} \right) = -\sum_{k=1}^{N} net_k \, \eta_\alpha (t_k - o_k) \, o_k^2 \tag{6}$$

where $\eta$ and $\eta_\alpha$ are learning parameters.

The learning algorithm initializes the centroid $W$ to the average of the patterns with target $1$, i.e., $W = \frac{1}{N_1} \sum_{k=1}^{N_1} I_k$, where $N_1$ is the number of patterns with target equal to $1$, and the tangent vectors to random vectors with small modulus. Then $\alpha$, the centroid $W$ and the tangent vectors $T_i$ are changed according to equations (4-6). Moreover, since the tangent vectors must constitute an ortho-normal basis, after each epoch of training the vectors $T_i$ are ortho-normalized.

## 5 The Constructive Algorithm

Before training the TD-neuron using equations (4-6), we have to set the tangent subspace dimension. The same problem is present in HSS and Diabolo (i.e., number of hidden units). To solve this problem we have developed a constructive algorithm which adds tangent vectors one by one according to the computational needs.

The key idea is based on the observation that a typical run of the learning algorithm described in Section 4 leads to the sequential convergence of the vectors according to their relative importance. This means that the tangent vectors all remain random vectors while the centroid converges first.

Then one of the tangent vectors converges to the most relevant transformation (while the remaining tangent vectors are still immature), and so on till all the tangent vectors converge, one by one, to less and less relevant transformations.

This behavior suggests starting the training using only the centroid (i.e., without tangent vectors) and allow it to converge. Then, as in other constructive algorithms, the centroid is frozen and one random tangent vector $T_1$ is added. Learning is resumed till changes in $T_1$ become irrelevant. During learning, however, $T_1$ is normalized after each epoch. At convergence, $T_1$ is frozen, a new random tangent vector $T_2$ is added, and learning is resumed. New tangent vectors are iteratively added till changes in the classification accuracy becomes irrelevant.

| # Tang. | HSS | | TD-neuron | | |
|---|---|---|---|---|---|
| | % Cor | % Err | % Cor | % Rej | % Err |
| 0 | — | — | 73.78 | 7.24 | 18.98 |
| 1 | 78.74 | 21.26 | 72.06 | 10.48 | 17.46 |
| 2 | 79.10 | 20.90 | 77.99 | 8.05 | 13.96 |
| 3 | 79.94 | 20.06 | 81.14 | 7.17 | 11.69 |
| 4 | 81.47 | 18.53 | 82.68 | 6.84 | 10.48 |
| 5 | 76.87 | 23.13 | 84.25 | 5.63 | 10.12 |
| 6 | 71.29 | 28.71 | 85.21 | 5.14 | 9.65 |
| 7 | — | — | 86.16 | 4.76 | 9.08 |
| 8 | — | — | 86.37 | 4.89 | 8.74 |

Table 1: The results obtained by the HSS algorithm and the TD-neuron.

# 6   Results

We have tested our constructive algorithm versus the HSS algorithm (which uses the 2-sided tangent distance) on 10587 binary digits from the NIST-3 dataset. The binary 128x128 digits were transformed into a 64-grey level 16x16 format by a simple local counting procedure. **No other pre-processing transformation was performed.** The training set consisted of 3000 randomly chosen digits, while the remaining digits where used in the test set. A single tangent model for each class of digit was computed using both algorithms. The classification of the test digits was performed using the label of the closest model for HSS and the output of the TD-neurons for our system. The TD-neurons used a rejection criterion with parameters adapted during training.

In Table 1 we have reported the performances on the test set of both HSS and our system. Different numbers of tangent vectors were tested for both of them. From the results it is clear that the models generated by HSS reach a peak in performance with 4 tangent vectors and then a sharp degradation of the generalization is observed by adding more tangent vectors. On the contrary, the TD-neurons are able to steadly increase the performance with an increasing number of tangent vectors. The improvement in the performance, however, seems to saturate when using many tangent vectors. Table 2 presents the confusion matrix obtained by the TD-neurons with 8 tangent vectors.

For comparison, we display some of the tangent models computed by HSS and by our algorithm in Figure 2. Note how tangent models developed by the HSS algorithm tend to be more blurred than the ones developed by our algorithm. This is due to the lake of discriminant capabilities by the HSS algorithm and it is the main cause of the degradation in performance observed when using more than 4 tangent vectors.

It must be pointed out that, for a fixed number of tangent vectors, the HSS algorithm is faster than ours, because it needs only a fraction of the training examples (only one class). However, our algorithm is remarkably more efficient when a family of tangent models with an increasing number of tangent vectors must be generated[2]. Moreover, since a TD-neuron uses the one sided tangent distance, it is faster in computing the output.

# 7   Conclusion

We introduced the *tangent distance neuron* (TD-neuron), which implements the 1-sided version of the tangent distance and gave a constructive learning algorithm for building a tangent subspace with discriminant capabilities. As stated in the in-

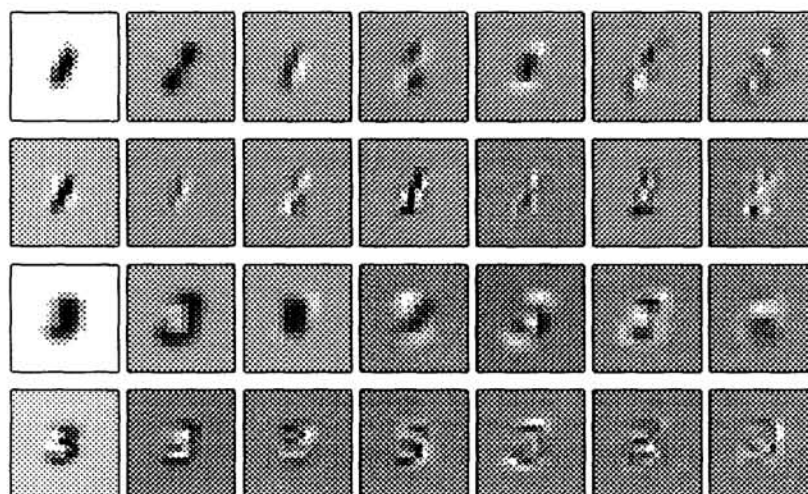

Figure 2: The tangent models obtained for digits '1' and '3' by the HSS algorithm (row 1 and 3, respectively) and our TD-neuron (row 2 and 4, respectively). The centroids are shown in the first column.

troduction, there are many advantages of using the proposed computational model versus other techniques like HSS and Diabolo. Specifically, we believe that the proposed approach is particularly useful in those applications where it is very important to have a classification system which is both discriminant and semantically transparent, in the sense that it is very easy to understand how it works. One among these applications is the classification of ancient book scripts. In fact, the description, the comparison, and the classification of forms are the main tasks of paleographers. Until now, however, these tasks have been generally performed without the aid of a universally accepted and quantitatively based method or technique. Consequently, very often it is impossible to reach a definitive date attribution of a document to within 50 years. In this field, it is very important to have a system which is both discriminant and explanatory, so that paleographers can learn from it which are the relevant features of the script of a given epoch. These requirements rule out systems like Diabolo, which is not easily interpretable, and also tangent models developed by HSS, which are not discriminant. In Figure 3 we have reported some preliminary results we obtained within this field.

Perhaps most importantly, our work suggests a number of research avenues. We used just a single TD-neuron; presumably having several neurons arranged as an adaptive pre-processing layer within a standard feed-forward neural network can yield a remarkable increase in the transformation invariant features of the network.

| | $o_0$ | $o_1$ | $o_2$ | $o_3$ | $o_4$ | $o_5$ | $o_6$ | $o_7$ | $o_8$ | $o_9$ | Rej | % Cor | % Rej | % Err |
|---|---|---|---|---|---|---|---|---|---|---|---|---|---|---|
| $C_0$ | 661 | 4 | 2 | 5 | 27 | 8 | 2 | 1 | 9 | 0 | 42 | 86.86 | 5.52 | 7.62 |
| $C_1$ | 4 | 842 | 0 | 1 | 1 | 1 | 1 | 11 | 8 | 0 | 9 | 95.90 | 1.03 | 3.08 |
| $C_2$ | 4 | 1 | 650 | 2 | 13 | 2 | 10 | 6 | 9 | 0 | 69 | 84.86 | 9.01 | 6.14 |
| $C_3$ | 1 | 3 | 22 | 656 | 0 | 26 | 1 | 11 | 18 | 4 | 28 | 85.19 | 3.64 | 11.17 |
| $C_4$ | 0 | 0 | 2 | 0 | 633 | 3 | 4 | 5 | 7 | 48 | 32 | 86.24 | 4.36 | 9.40 |
| $C_5$ | 1 | 1 | 3 | 39 | 2 | 535 | 7 | 1 | 7 | 3 | 44 | 83.20 | 6.84 | 9.95 |
| $C_6$ | 0 | 4 | 1 | 2 | 6 | 11 | 680 | 0 | 4 | 0 | 33 | 91.77 | 4.45 | 3.78 |
| $C_7$ | 1 | 1 | 3 | 0 | 4 | 3 | 0 | 727 | 12 | 24 | 27 | 90.65 | 3.37 | 5.99 |
| $C_8$ | 1 | 4 | 7 | 18 | 14 | 12 | 0 | 7 | 607 | 11 | 62 | 81.70 | 8.34 | 9.96 |
| $C_9$ | 0 | 0 | 0 | 12 | 43 | 1 | 0 | 70 | 36 | 562 | 25 | 75.03 | 3.34 | 21.63 |
| Total | | | Correct: | 86.37% | | | Rejected | 4.89% | | | | Errors | 8.74% | |

Table 2: The confusion matrix for the TD-neurons with 8 tangent vectors.

## Footnotes

[1] A special case of tangent distance, i.e., the one sided tangent distance $D_T^{1-sided}(X_i, X_j) = \min_{\theta_i} \|X_i(\theta_i) - X_j\|$, can be computed more efficiently [SS95].

[2]The tangent model computed by HSS depends on the number of tangent vectors.
